# Optimizing Cortical Mappings

**Geoffrey J. Goodhill**
The Salk Institute
10010 North Torrey Pines Road
La Jolla, CA 92037, USA

**Steven Finch**
Human Communication Research Centre
University of Edinburgh, 2 Buccleuch Place
Edinburgh EH8 9LW, GREAT BRITAIN

**Terrence J. Sejnowski**
The Howard Hughes Medical Institute
The Salk Institute for Biological Studies
10010 North Torrey Pines Road, La Jolla, CA 92037, USA
&
Department of Biology, University of California San Diego
La Jolla, CA 92037, USA

## Abstract

"Topographic" mappings occur frequently in the brain. A popular approach to understanding the structure of such mappings is to map points representing input features in a space of a few dimensions to points in a 2 dimensional space using some self-organizing algorithm. We argue that a more general approach may be useful, where similarities between features are not constrained to be geometric distances, and the objective function for topographic matching is chosen explicitly rather than being specified implicitly by the self-organizing algorithm. We investigate analytically an example of this more general approach applied to the structure of interdigitated mappings, such as the pattern of ocular dominance columns in primary visual cortex.

## 1 INTRODUCTION

A prevalent feature of mappings in the brain is that they are often "topographic". In the most straightforward case this simply means that neighbouring points on a two-dimensional sheet (e.g. the retina) are mapped to neighbouring points in a more central two-dimensional structure (e.g. the optic tectum). However a more complex case, still often referred to as topographic, is the mapping from an abstract space of features (e.g. position in the visual field, orientation, eye of origin etc) to

the cortex (e.g. layer 4 of V1). In many cortical sensory areas, the preferred sensory stimuli of neighbouring neurons changes slowly, except at discontinuous jumps, suggestive of an optimization principle that attempts to match "similar" features to nearby points in the cortex. In this paper, we (1) discuss what might constitute an appropriate measure of similarity between features, (2) outline an optimization principle for matching the similarity structure of two abstract spaces (i.e. a measure of the degree of topography of a mapping), and (3) use these ideas to analyse the case where two equivalent input variables are mapped onto one target structure, such as the "ocular dominance" mapping from the right and left eyes to V1 in the cat and monkey.

## 2   SIMILARITY MEASURES

A much-investigated computational approach to the study of mappings in V1 is to consider the input features as points in a multidimensional euclidean space [1, 5, 9]. The input dimensions then consist of e.g. spatial position, orientation, ocular dominance, and so on. Some distribution of points in this space is assumed which attempts, in some sense, to capture the statistics of these features in the visual world. For instance, in [5], distances between points in the space are interpreted as a decreasing function of the degree to which the corresponding features are correlated over an ensemble of images. Some self-organizing algorithm is then applied which produces a mapping from the high-dimensional feature space to a two-dimensional sheet representing the cortex, such that nearby points in the feature space map to nearby points in the two-dimensional sheet.[1]

However, such approaches assume that the dissimilarity structure of the input features is well-captured by euclidean distances in a geometric space. There is no particular reason why this should be true. For instance, such a representation implies that the dissimilarity between features can become arbitrarily large, an unlikely scenario. In addition, it is difficult to capture higher-order relationships in such a representation, such as that two oriented line-segment detectors will be more correlated if the line segments are co-linear than if they are not. We propose instead that, for a set of features, one could construct directly from the statistics of natural stimuli a feature *matrix* representing similarities or dissimilarities, without regard to whether the resulting relationships can be conveniently captured by distances in a euclidean feature space. There are many ways this could be done; one example is given below. Such a similarity matrix for features can then be optimally matched (in some sense) to a similarity matrix for positions in the output space.

A disadvantage from a computational point of view of this generalized approach is that the self-organizing algorithms of e.g. [6, 2] can no longer be applied, and possibly less efficient optimization techniques are required. However, an advantage of this is that one may now explore the consequences of optimizing a whole range of objective functions for quantifying the quality of the mapping, rather than having to accept those given explicitly or implicitly by the particular self-organizing algorithm.

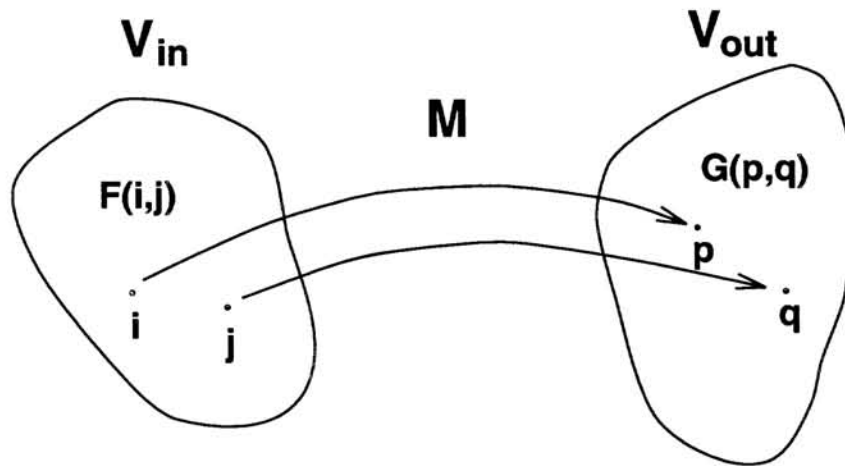

Figure 1: The mapping framework.

## 3   OPTIMIZATION PRINCIPLES

We now outline a general framework for measuring to what degree a mapping matches the structure of one similarity matrix to that of another. It is assumed that input and output matrices are of the same (finite) dimension, and that the mapping is bijective. Consider an input space $V_{in}$ and an output space $V_{out}$, each of which contains N points. Let M be the mapping from points in $V_{in}$ to points in $V_{out}$ (see figure 1). We use the word "space" in a general sense: either or both of $V_{in}$ and $V_{out}$ may not have a geometric interpretation. Assume that for each space there is a symmetric "similarity" function which, for any given pair of points in the space, specifies how similar (or dissimilar) they are. Call these functions F for $V_{in}$ and G for $V_{out}$. Then we define a cost functional C as follows

$$C = \sum_{i=1}^{N} \sum_{j<i} F(i,j)G(M(i), M(j)),    \tag{1}$$

where i and j label points in $V_{in}$, and M(i) and M(j) are their respective images in $V_{out}$. The sum is over all possible pairs of points in $V_{in}$. Since M is a bijection it is invertible, and C can equivalently be written

$$C = \sum_{i=1}^{N} \sum_{j<i} F(M^{-1}(i), M^{-1}(j))G(i,j),    \tag{2}$$

where now i and j label points in $V_{out}$, and $M^{-1}$ is the inverse map. A good (i.e. highly topographic) mapping is one with a high value of C. However, if one of F or G were given as a dissimilarity function (i.e. increasing with decreasing similarity) then a good mapping would be one with a low value of C. How F and G are defined is problem-specific.

C has a number of important properties that help to justify its adoption as a measure of the degree of topography of a mapping (for more details see [3]). For instance, it can be shown that if a mapping that *preserves ordering relationships* between two similarity matrices exists, then maximizing C will find it. Such maps are homeomorphisms. However not all homeomorphisms have this property, so we refer to such "perfect" maps as "topographic homeomorphisms". Several previously defined optimization principles, such as minimum path and minimum

wiring [1], are special cases of C. It is also closely related (under the assumptions above) to Luttrell's minimum distortion measure [7], if F is euclidean distance in a geometric input space, and G gives the noise process in the output space.

## 4  INTERDIGITATED MAPPINGS

As a particular application of the principles discussed so far, we consider the case where the similarity structure of $V_{in}$ can be expressed in matrix form as

$$\left( \begin{array}{cc} Q_S & Q_C \\ Q_C & Q_S \end{array} \right)$$

where $Q_S$ and $Q_C$ are of dimension $N/2$. This means that $V_{in}$ consists of two halves, each with the same internal similarity structure, and an in general different similarity structure between the two halves. The question is how best to match this dual similarity structure to a single similarity structure in $V_{out}$. This is of mathematical interest since it is one of the simplest cases of a mismatch between the similarity structures of $V_{in}$ and $V_{out}$, and of biological interest since it abstractly represents the case of input from two equivalent sets of receptors coming together in a single cortical sheet, e.g. ocular dominance columns in primary visual cortex (see e.g. [8, 5]). For simplicity we consider only the case of two one-dimensional retinae mapping to a one-dimensional cortex.

The feature space approach to the problem presented in [5] says that the dissimilarities in $V_{in}$ are given by squared euclidean distances between points arranged in two parallel rows in a two-dimensional space. That is,

$$F(i,j) = \begin{cases} |i-j|^2 & : & i,j \text{ in same half of } V_{in} \\ |i-j-N/2|^2 + k^2 & : & i,j \text{ in different halves of } V_{in} \end{cases} \qquad (3)$$

assuming that indices $1 \ldots N/2$ give points in one half and indices $N/2+1 \ldots N$ give points in the other half. $G(i,j)$ is given by

$$G(i,j) = \begin{cases} 1 & : & i,j \text{ neighbouring} \\ 0 & : & \text{otherwise} \end{cases} \qquad (4)$$

It can be shown that the globally optimal mapping (i.e. minimum of C) when $k > 1$ is to keep the two halves of $V_{in}$ entirely separate in $V_{out}$ [5]. However, there is also a local minimum for an interdigitated (or "striped") map, where the interdigitations have width $n = 2k$. By varying the value of $k$ it is thus possible to smoothly vary the periodicity of the locally optimal striped map. Such behavior predicted the outcome of a recent biological experiment [4]. For $k < 1$ the globally optimal map is stripes of width $n = 2$.

However, in principle many alternative ways of measuring the similarity in $V_{in}$ are possible. One obvious idea is to assume that similarity is given directly by the degree of correlation between points within and between the two eyes. A simple assumption about the form of these correlations is that they are a gaussian function of physical distance between the receptors (as in [8]). That is,

$$F(i,j) = \begin{cases} e^{-\alpha|i-j|^2} & : & i,j \text{ in same half of } V_{in} \\ ce^{-\beta|i-j-N/2|^2} & : & i,j \text{ in different halves of } V_{in} \end{cases} \qquad (5)$$

with $c < 1$. We assume for ease of analysis that G is still as given in equation 4. This directly implements an intuitive notion put forward to account for the interdigitation of the ocular dominance mapping [4]: that the cortex tries to represent

similar inputs close together, that similarity is given by the degree of correlation between the activities of points (cells), and additionally that natural visual scenes impose a correlational structure of the same qualitative form as equation 5. We now calculate C analytically for various mappings (c.f. [5]), and compare the cost of a map that keeps the two halves of $V_{in}$ entirely separate in $V_{out}$ to those which interdigitate the two halves of $V_{in}$ with some regular periodicity. The map of the first type we consider will be refered to as the "up and down" map: moving from one end of $V_{out}$ to the other implies moving entirely through one half of $V_{in}$, then back in the opposite direction through the other half. For this map, the cost $C_{ud}$ is given by

$$C_{ud} = 2(N-1)e^{-\alpha} + c. \tag{6}$$

For an interdigitated (striped) map where the stripes are of width $n \geq 2$:

$$C_s(n) = N\left[2\left(1 - \frac{1}{n}\right)e^{-\alpha} + \frac{c}{n}\left(e^{-\beta f(n)} + e^{-\beta g(n)}\right)\right] \tag{7}$$

where for $n$ even $f(n) = g(n) = \left(\frac{n-2}{2}\right)^2$ and for $n$ odd $f(n) = \left(\frac{n-1}{2}\right)^2$, $g(n) = \left(\frac{n-3}{2}\right)^2$. To characterize this system we now analyze how the $n$ for which $C_s(n)$ has a local maximum varies with $c, \alpha, \beta$, and when this local maximum is also a global maximum. Setting $\frac{dC_s(n)}{dn} = 0$ does not yield analytically tractable expressions (unlike [5]). However, more direct methods can be used: there is a local maximum at $n$ if $C_s(n-1) < C_s(n) > C_s(n+1)$. Using equation 7 we derive conditions on $c$ for this to be true. For $n$ odd, we obtain the condition $c_1 < c < c_2$ where $c_1 = c_2$; that is, there are no local maxima at odd values of $n$. For $n$ even, we also obtain $c_1 < c < c_2$ where now

$$c_1 = \frac{2e^{-\alpha}}{ne^{-\beta\left(\frac{n-4}{2}\right)^2} - (n-2)e^{-\beta\left(\frac{n-2}{2}\right)^2}}$$

and $c_2(n) = c_1(n+2)$. $c_1(n)$ and $c_2(n)$ are plotted in figure 2, from which one can see the ranges of $c$ for which particular $n$ are local maxima. As $\beta$ increases, maxima for larger values of $n$ become apparent, but the range of $c$ for which they exist becomes rather small. It can be shown that $C_{ud}$ is always the global maximum, except when $e^{-\alpha} > c$, when $n = 2$ is globally optimal. As $c$ decreases the optimal stripe width gets wider, analogously to $k$ increasing in the dissimilarities given by equation 3. When $\beta$ is such that there is no local maximum the only optimum is stripes as wide as possible. This fits with the intuitive idea that if corresponding points in the two halves of $V_{in}$ (i.e. $|i - j| = N/2$) are sufficiently similar then it is favorable to interdigitate the two halves in $V_{out}$, otherwise the two halves are kept completely separate.

The qualitative behavior here is similar to that for equation 3. $n = 2$ is a global optimum for large $c$ (small $k$), then as $c$ decreases ($k$ increases) $n = 2$ first becomes a local optimum, then the position of the local optimum shifts to larger $n$. However, an important difference is that in equation 3 the dissimilarities increase without limit with distance, whereas in equation 5 the similarities tend to zero with distance. Thus for equation 5 the extra cost of stripes one unit wider rapidly becomes negligible, whereas for equation 3 this extra cost keeps on increasing by ever larger amounts. As $n \to \infty$, $C_{ud} \sim C_s(n)$ for the similarities defined by equation 5 (i.e. there is the same cost for traversing the two blocks in the same direction as in the opposite direction), whereas for the dissimilarities defined by equation 3 there is a quite different cost in these two cases. That F and G should tend to a bounded value as $i$ and $j$ become ever more distant neighbors seems biologically more plausible than that they should be potentially unbounded.

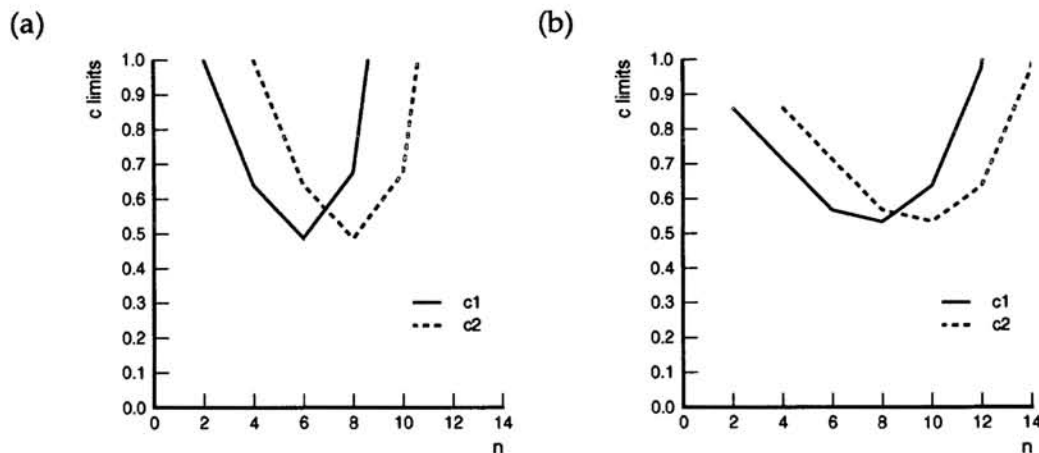

Figure 2: The ranges of c for which particular $n$ are local maxima. (a) $\alpha = \beta = 0.25$. (b) $\alpha = 0.25$, $\beta = 0.1$. When the $c_2$ (dashed) line is below the $c_1$ (solid) line no local maxima exist. For each (even) value of $n$ to the left of the crossing point, the vertical range between the two lines gives the values of c for which that $n$ is a local maximum. Below the solid line and to the right of the crossing point the only maximum is stripes as wide as possible.

Issues such as those we have addressed regarding the transition from "striped" to "blocked" solutions for combining two sets of inputs distinguished by their intra- and inter-population similarity structure may be relevant to understanding the spatial representation of functional attributes across cortex. The results suggest the hypothesis that two variables are interdigitated in the same area rather than being represented separately in two distinct areas if the inter-population similarity is sufficiently high. An interesting point is that the striped solutions are often only local optima. It is possible that in reality developmental constraints (e.g. a chemically defined bias towards overlaying the two projections) impose a bias towards finding a striped rather than blocked solution, even though the latter may be the global optimum.

## 5   DISCUSSION

We have argued that, in order to understand the structure of mappings in the brain, it could be useful to examine more general measures of similarity and of topographic matching than those implied by standard feature space models. The consequences of one particular alternative set of choices has been examined for the case of an interdigitated map of two variables. Many alternative objective functions for topographic matching are of course possible; this topic is reviewed in [3]. Two issues we have not discussed are the most appropriate way to define the features of interest, and the most appropriate measures of similarity between features (see [10] for an interesting discussion).

A next step is to apply these methods to more complex structures in V1 than just the ocular dominance map. By examining more of the space of possibilities than that occupied by the current feature space models, we hope to understand more about the optimization strategies that might be being pursued by the cortex. Feature space models may still turn out to be more or less the right answer; however even if this is true, our approach will at least give a deeper level of understanding why.

## Acknowledgements

We thank Gary Blasdel, Peter Dayan and Paul Viola for stimulating discussions.

## Footnotes

[1]We mean this in a rather loose sense, and wish to include here the principles of mapping nearby points in the sheet to nearby points in the feature space, mapping distant points in the feature space to distant points in the sheet, and so on.

## References

[1] Durbin, R. & Mitchison, G. (1990). A dimension reduction framework for understanding cortical maps. *Nature*, **343**, 644-647.

[2] Durbin, R. & Willshaw, D.J. (1987). An analogue approach to the travelling salesman problem using an elastic net method. *Nature*, **326**, 689-691.

[3] Goodhill, G. J., Finch, S. & Sejnowski, T. J. (1995). Quantifying neighbourhood preservation in topographic mappings. Institute for Neural Computation Technical Report Series, No. INC-9505, November 1995. Available from ftp://salk.edu/pub/geoff/goodhill_finch_sejnowski_tech95.ps.Z or http://cnl.salk.edu/~geoff.

[4] Goodhill, G.J. & Löwel, S. (1995). Theory meets experiment: correlated neural activity helps determine ocular dominance column periodicity. *Trends in Neurosciences*, **18**, 437-439.

[5] Goodhill, G.J. & Willshaw, D.J. (1990). Application of the elastic net algorithm to the formation of ocular dominance stripes. *Network*, **1**, 41-59.

[6] Kohonen, T. (1982). Self-organized formation of topologically correct feature maps. *Biol. Cybern.*, **43**, 59-69.

[7] Luttrell, S.P. (1990). Derivation of a class of training algorithms. *IEEE Trans. Neural Networks*, **1**, 229-232.

[8] Miller, K.D., Keller, J.B. & Stryker, M.P. (1989). Ocular dominance column development: Analysis and simulation. *Science*, **245**, 605-615.

[9] Obermayer, K., Blasdel, G.G. & Schulten, K. (1992). Statistical-mechanical analysis of self-organization and pattern formation during the development of visual maps. *Phys. Rev. A*, **45**, 7568-7589.

[10] Weiss, Y. & Edelman, S. (1995). Representation of similarity as a goal of early sensory coding. *Network*, **6**, 19-41.